# Experimental Evaluation of Learning in a Neural Microsystem

Joshua Alspector  Anthony Jayakumar  Stephan Luna†
Bellcore
Morristown, NJ 07962-1910

## Abstract

We report learning measurements from a system composed of a cascadable learning chip, data generators and analyzers for training pattern presentation, and an X-windows based software interface. The 32 neuron learning chip has 496 adaptive synapses and can perform Boltzmann and mean-field learning using separate noise and gain controls. We have used this system to do learning experiments on the parity and replication problem. The system settling time limits the learning speed to about 100,000 patterns per second roughly independent of system size.

## 1. INTRODUCTION

We have implemented a model of learning in neural networks using feedback connections and a local learning rule. Even though back-propagation[1] (Rumelhart,1986) networks are feedforward in processing, they have separate, implicit feedback paths during learning for error propagation. Networks with explicit, full-time feedback paths can perform pattern completion[2] (Hopfield,1982), can learn many-to-one mappings, can learn probability distributions, and can have interesting temporal and dynamical properties in contrast to the single forward pass processing of multilayer perceptrons trained with back-propagation or other means. Because of the potential for complex dynamics, feedback networks require a reliable method of relaxation for learning and retrieval of static patterns. The Boltzmann machine[3] (Ackley,1985) uses stochastic settling while the mean-field theory version[4] (Peterson,1987) uses a more computationally efficient deterministic technique.

We have previously shown that Boltzmann learning can be implemented in VLSI[5] (Alspector,1989). We have also shown, by simulation,[6] (Alspector,1991a) that Boltzmann and mean-field networks can have powerful learning and representation properties just like the more thoroughly studied back-propagation methods. In this paper, we demonstrate these properties using new, expandable parallel hardware for on-chip learning.

## 2. VLSI IMPLEMENTATION

### 2.1 Electronic Model

We have implemented these feedback networks in VLSI which speeds up learning by many orders of magnitude due to the parallel nature of weight adjustment and neuron state update. Our choice of learning technique for implementation is due mainly to the local learning rule which makes it much easier to cast these networks into electronics than back-propagation.

Individual neurons in the Boltzmann machine have a probabilistic decision rule such that neuron $i$ is in state $s_i = 1$ with probability

$$Pr(s_i = 1) = \frac{1}{1+e^{-u_i/T}} \qquad (1)$$

where $u_i = \sum_j w_{ij} s_j$ is the net input to each neuron calculated by current summing and $T$ is a parameter that acts like temperature in a physical system and is represented by the noise and gain terms in Eq. (2), which follows. In the electronic model we use, each neuron performs the activation computation

$$s_i = f\left(\beta * (u_i + v_i)\right) \qquad (2)$$

where $f$ is a monotonic non-linear function such as *tanh*. The noise, $v$, is chosen from a zero mean gaussian distribution whose width is proportional to the temperature. This closely approximates the distribution in Eq. (1) and comes from our hardware implementation, which supplies uncorrelated noise in the form of a binomial distribution[7] (Alspector,1991b) to each neuron. The noise is slowly reduced as annealing proceeds. For mean-field learning, the noise is zero but the gain, $\beta$, has a finite value proportional to $1/T$ taken from the annealing schedule. Thus the non-linearity sharpens as 'annealing' proceeds.

The network is annealed in two phases, + and −, corresponding to clamping the outputs in the desired state (teacher phase) and allowing them to run free (student phase) at each pattern presentation. The learning rule which adjusts the weights $w_{ij}$ from neuron $j$ to neuron $i$ is

$$\Delta w_{ij} = sgn\left[ (s_i s_j)^+ - (s_i s_j)^- \right]. \qquad (3)$$

Note that this measures the instantaneous correlations after annealing. For both phases each synapse memorizes the correlations measured at the end of the annealing cycle and weight adjustment is then made, (i.e., online). The *sgn* matches our hardware implementation which changes weights by one each time.

### 2.2 Learning Microchip

Fig. 1 shows the learning microchip which has been fabricated. It contains 32 neurons and 992 connections (496 bidirectional synapses). On the extreme right is a noise generator which supplies 32 uncorrelated pseudo-random noise sources[7] (Alspector,1991b) to the neurons to their left. These noise sources are summed in the form of current along with the weighted post-synaptic signals from other neurons at the input to each neuron in order to implement the simulated annealing process of the stochastic Boltzmann machine. The neuron amplifiers implement a non-linear activation

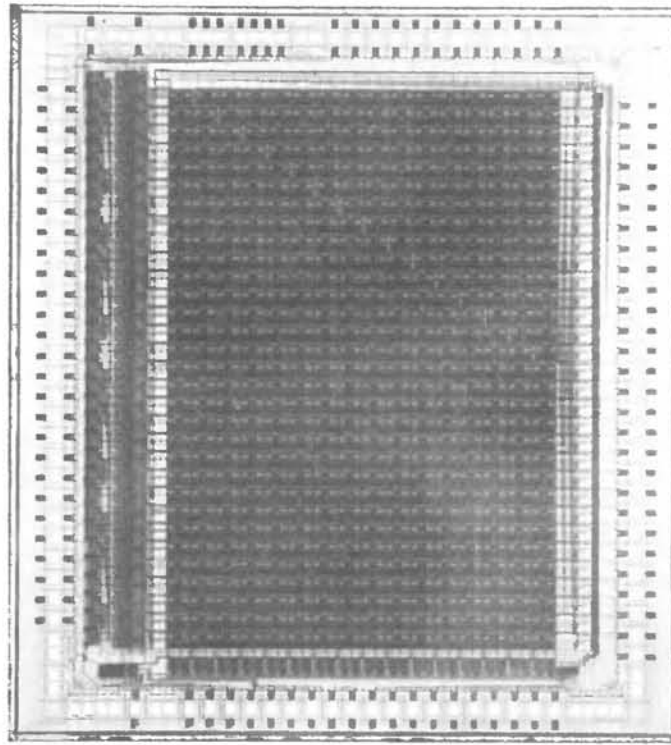

Figure 1. Photo of 32-Neuron Cascadable Learning Chip

function which has variable gain to provide for the gain sharpening function of the mean-field technique. The range of neuron gain can also be adjusted to allow for scaling in summing currents due to adjustable network size.

Most of the area is occupied by the synapse array. Each synapse digitally stores a weight ranging from -15 to +15 as 4 bits plus a sign. It multiples the voltage input from the presynaptic neuron by this weight to output a current. One conductance direction can be disconnected so that we can experiment with asymmetric networks[8] (Allen,1990). Although the synapses can have their weights set externally, they are designed to be adaptive. They store correlations, in parallel, using the local learning rule of Eq. (3) and adjust their weights accordingly. A neuron state range of -1 to 1 is assumed by the digital learning processor in each synapse on the chip.

Fig. 2a shows a family of transfer functions of a neuron, showing how the gain is continually adjustable by varying a control voltage. Fig. 2b shows the transfer function of a synapse as different weights are loaded. The input linear range is about 2 volts.

Fig. 3 shows waveforms during exclusive-OR learning using the noise annealing of the Boltzmann machine. The top three traces are hidden neurons while the bottom trace is the output neuron which is clamped during the + phase. There are two input patterns presented during the time interval displayed, (-1,+1) and (+1,-1), both of which should output a +1 (note the state clamped to high voltage on the output neuron). Note the sequence of steps involved in each pattern presentation. 1) Outputs from the previous pattern are unclamped. 2) The new pattern is presented to the input neurons. 3) Noise is presented to the network and annealed. 4) The student phase latch captures the

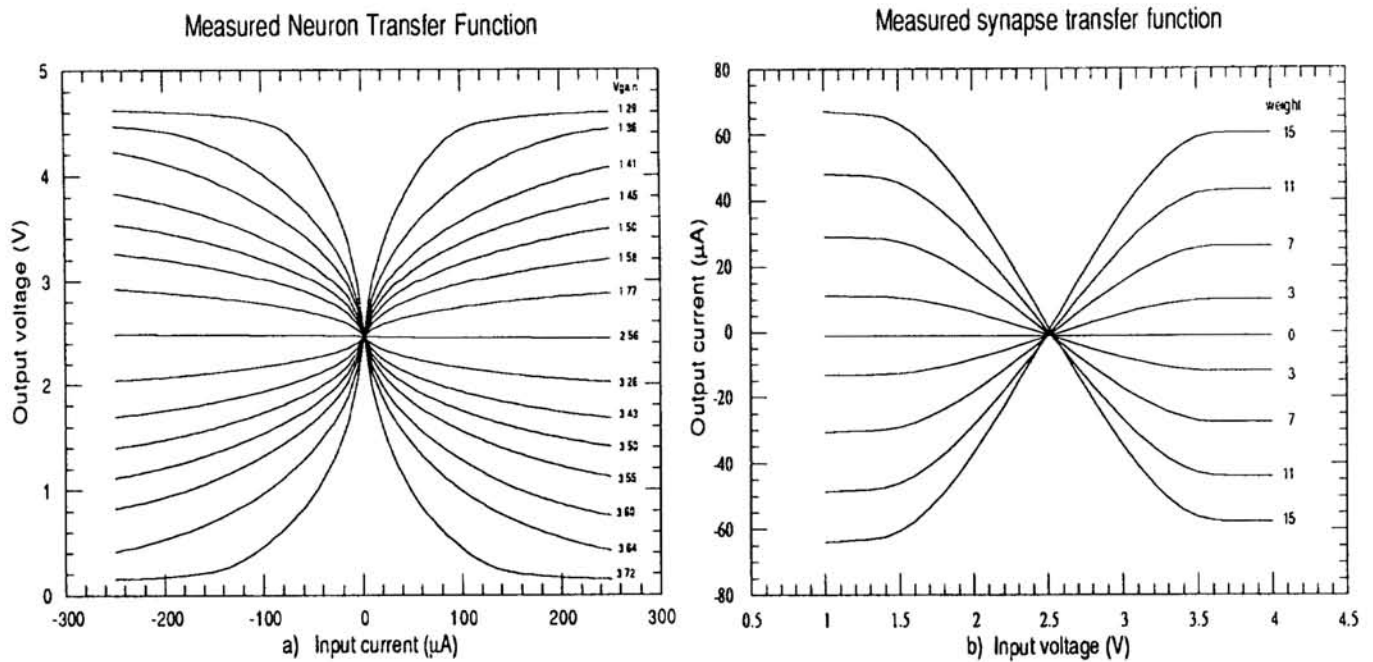

Figure 2.  Transfer Functions of Electronic Neuron (2a) and Synapse (2b)

correlations.  5) Data from the neuron states is read into the data analyzer.  6) The output neurons are clamped (no annealing is necessary for a three layer network).  7) The teacher phase latch captures the correlations.  8) Weights are adjusted (go to step 1).

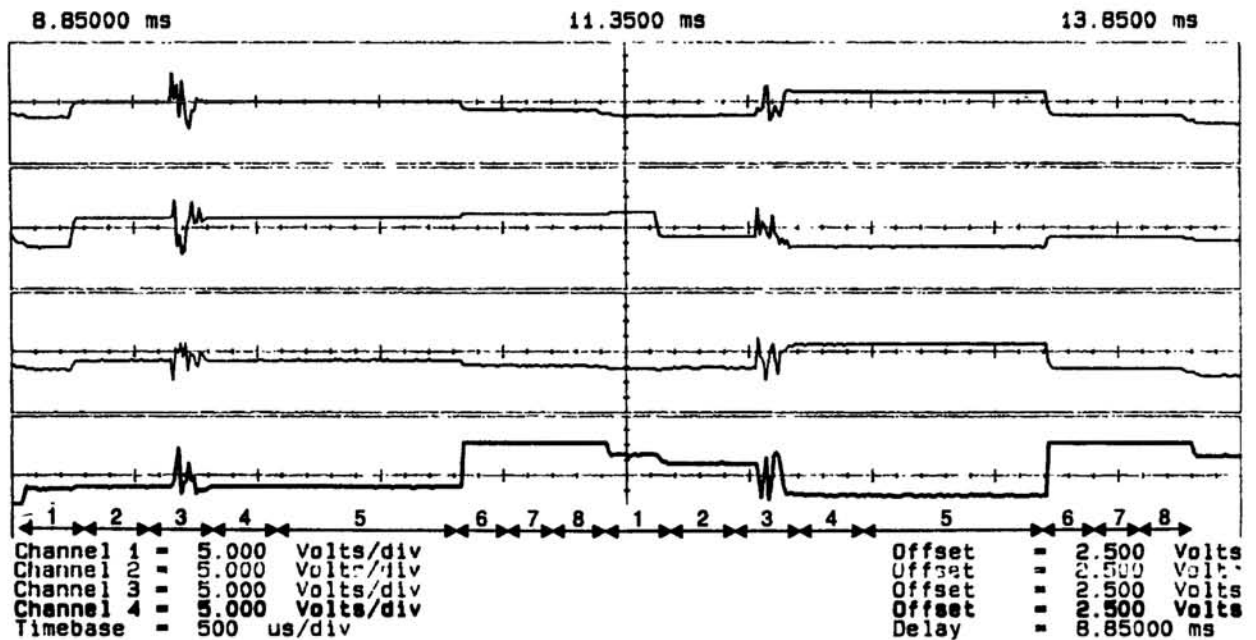

Figure 3.  Neuron Signals during Learning (see text for steps involved)

Fig. 4a shows an expanded view of 4 neuron waveforms during the noise annealing portion of the chip operation during Boltzmann learning.  Fig. 4b shows a similar portion during gain annealing.  Note that, at low gain, the neuron states start at 2.5 *volts* and settle to an analog value between 0 and 5 *volts*.  For the purposes of classification for the

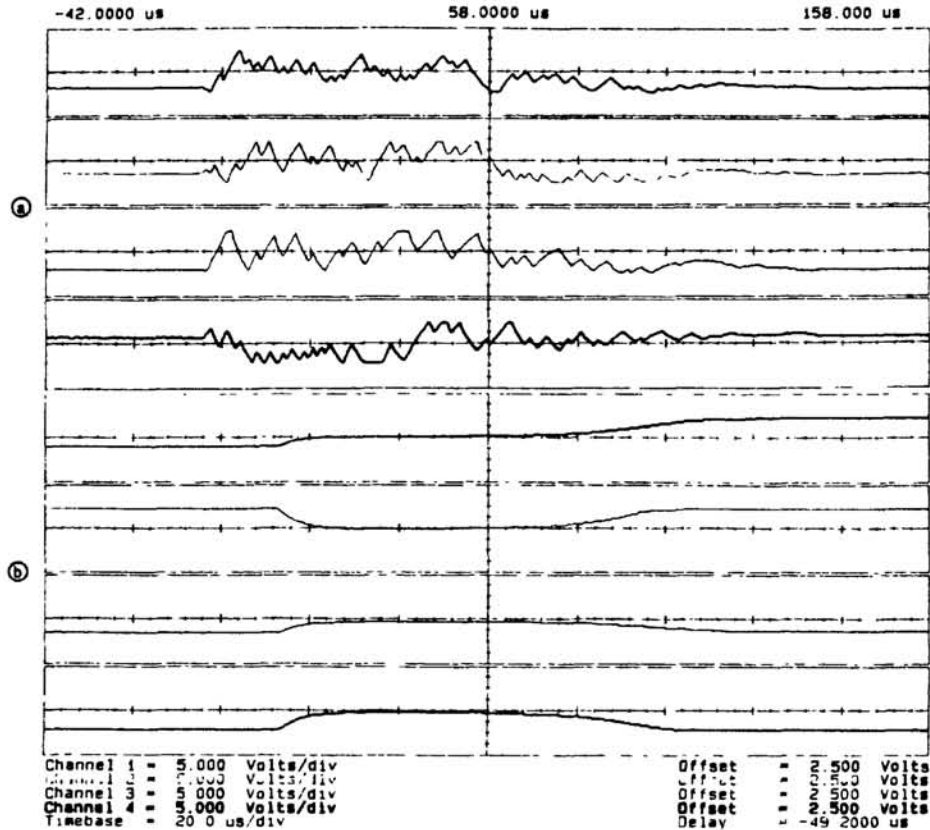

Figure 4. Neuron Signals during Annealing with Noise (4a) and Gain (4b)

digital problems we investigated, neurons are either +1 or -1 depending on whether their voltage is above or below 2.5 *volts*. This isn't clear until after settling. There are several instances in Figs. 3 and 4 where the neuron state changes after noise or gain annealing.

The speed of pattern presentations is limited by the length of the annealing signal for system settling (100 μsec in Fig. 3). The rest of the operations can be made negligibly short in comparison. The annealing time could be reduced to 10 μsec or so, leading to a rate of about 100,000 patterns/sec. In comparison, a 10-10-10 replication problem, which fits on a single chip, takes about a second per pattern on a SPARCstation 2. This time scales roughly with the number of weights on a sequential machine, but is almost constant on the learning chip due to its parallel nature.

We can do even larger problems in a multiple chip system because the chip is designed to be cascaded with other similar chips in a board-level system which can be accessed by a computer. The nodes which sum current from synapses for net input into a neuron are available externally for connection to other chips and for external clamping of neurons or other external input. We are currently building such a system with a VME bus interface for tighter coupling to our software than is allowed by the GPIB instrument bus we are using at the time of this writing.

### 2.3 Learning Experiments

To study learning as a function of problem size, we chose the parity and replication (identity) problems. This facilitates comparisons with our previous simulations[6]

(Alspector,1991a). The parity problem is the generalization of exclusive-OR for arbitrary input size. It is difficult because the classification regions are disjoint with every change of input bit, but it has only one output. The goal of the replication problem is for the output to duplicate the bit pattern found on the input after being encoded by the hidden layer. Note that the output bits can be shifted or scrambled in any order without affecting the difficulty of the problem. There are as many output neurons as input. For the replication problem, we chose the hidden layer to have the same number of neurons as the input layer, while for parity we chose the hidden layer to have twice the number as the input layer.

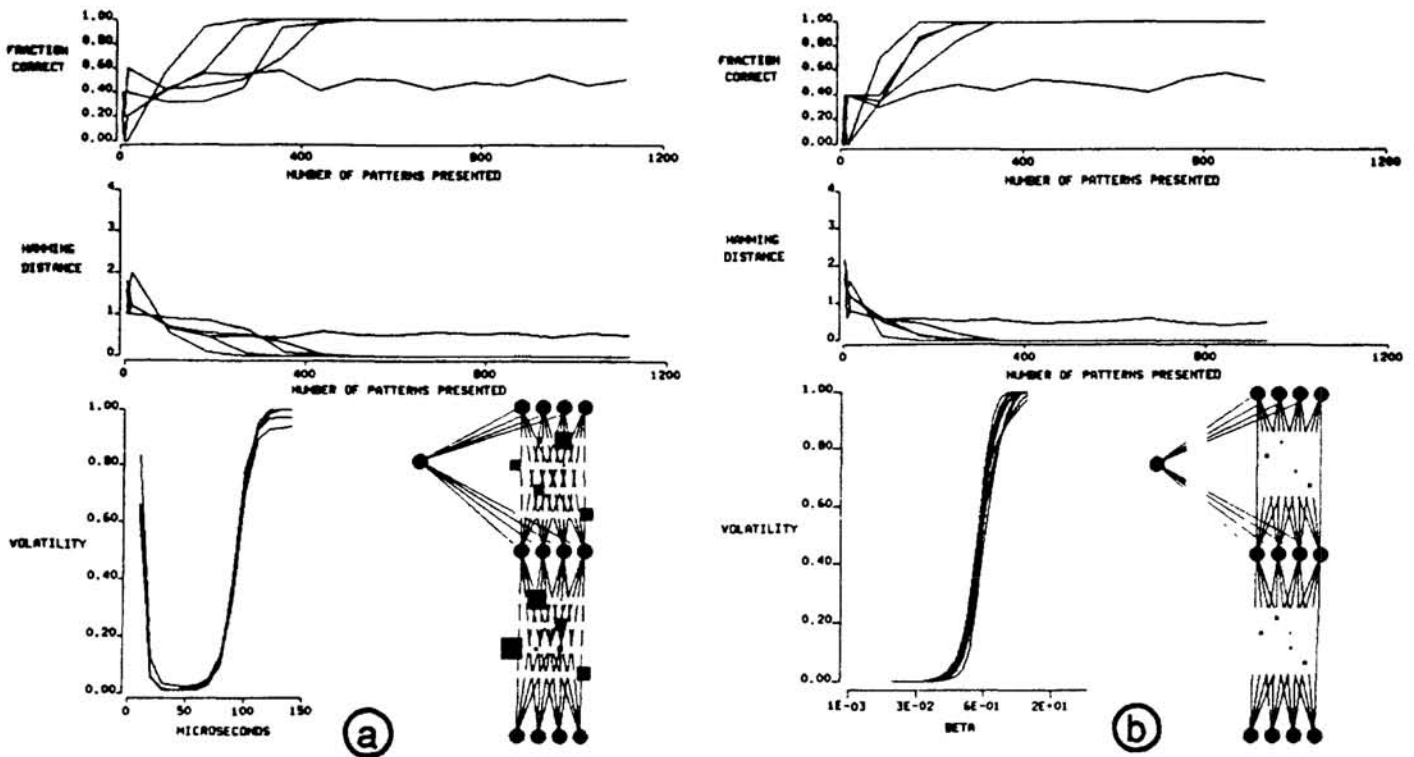

Figure 5. X-window Display for Learning on Chip (5a) and in Software (5b)

Fig. 5 shows the X-window display for 5 mean-field runs for learning the 4 input, 4 hidden, 4 output (4-4-4) replication on the chip (5a) and in the simulator (5b). The user specification is the same for both. Only the learning calculation module is different. Both have displays of the network topology, the neuron states (color and pie-shaped arc of circles) and the network weights (color and size of squares). There are also graphs of percent correct and error (Hamming distance for replication) and one of volatility of neuron states[9] (Alspector,1992) as a measure of the system temperature. The learning curves look quite similar. In both cases, one of the 5 runs failed to learn to 100 %. The boxes representing weights are signed currents (about 4 $\mu amp$ per unit weight) in 5a and integers from -15 to +15 in 5b. Volatility is plotted as a function of time ($\mu sec$) in 5a and shows that, in hardware (see Fig. 4), time is needed for a gain decrease at the start of the annealing as well as for the gain increase of the annealing proper. The volatility in 5b is

plotted as a function of gain (BETA) which increases logarithmically in the simulator at each anneal step.

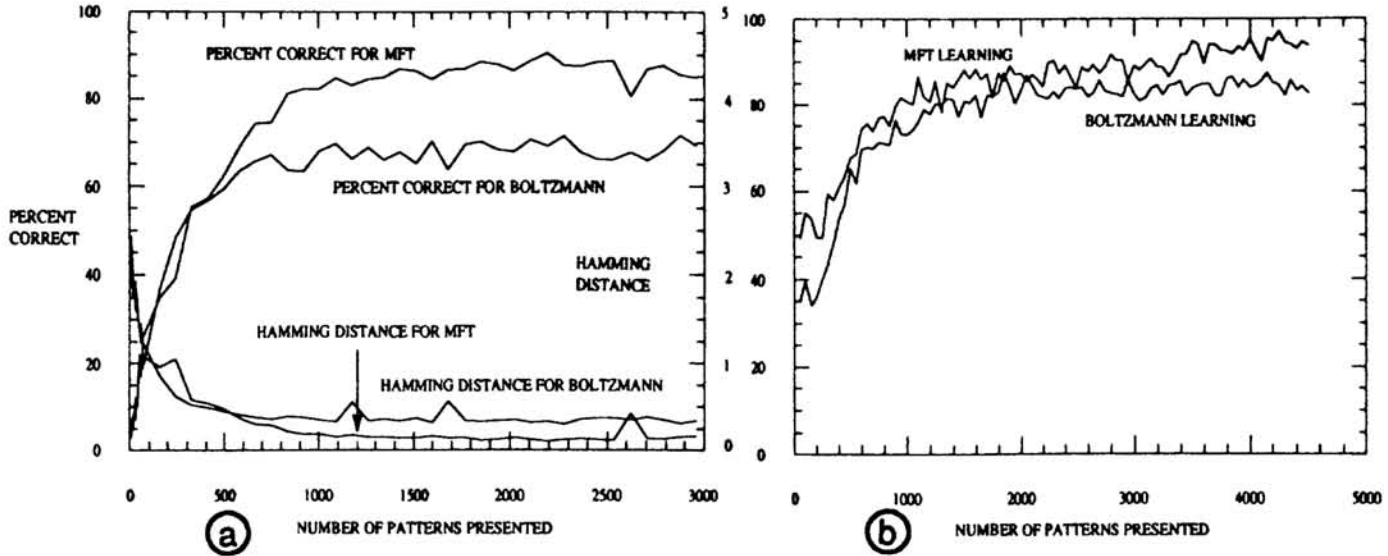

Figure 6. On-chip Learning for 6 Input Replication (6a) and Parity (6b)

Fig. 6a displays data from the average of 10 runs of 6-6-6 replication for both Boltzmann (BZ) and mean-field (MFT) learning. While the percent correct saturates at 90 % (70 % for Boltzmann), the output error as measured by the Hamming distance between input and output is less than 1 bit out of 6. Boltzmann learning is somewhat poorer in this experiment probably because circuit parameters have not yet been optimized. We expect that a combination of noise and gain annealing will yield the best results but have not tested this possibility at this writing. Fig. 6b is a similar plot for 6-12-1 parity.

We have done on-chip learning experiments using noise and gain annealing for parity and replication up to 8 input bits, nearly utilizing all the neurons on a single chip. To judge scaling behavior in these early experiments, we note the number of patterns required until no further improvement in percent correct is visible by eye. Fig. 7a plots, for an average of 10 runs of the parity problem, the number of patterns required to learn up to the saturation value for percent correct for both Boltzmann and mean-field learning. This scales roughly as an exponential in number of inputs for learning on chip just as it did in simulation[6] (Alspector,1991a) since the training set size is exponential. The final percent correct is indicated on the plot. Fig. 7b plots the equivalent data for the replication problem. Outliers are due to low saturation values. Overall, the training time per pattern on-chip is quite similar to our simulations. However, in real-time, it can be about 100,000 times as fast for a single chip and will be even faster for multiple chip systems. The speed for either learning or evaluation is roughly $10^8$ connections per second per chip.

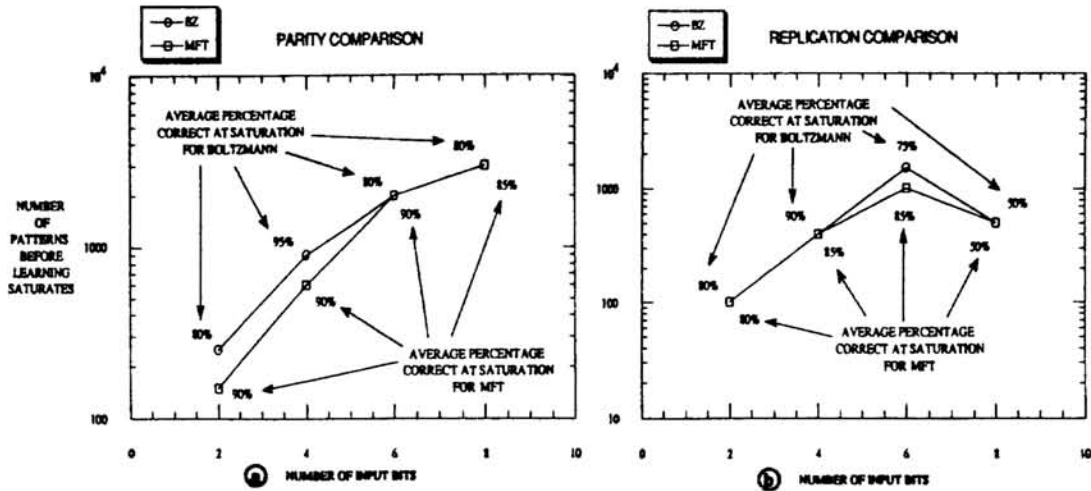

Figure 7. Scaling of Parity (7a) and Replication (7b) Problem with Input Size

## 3. CONCLUSION

We have shown that Boltzmann and mean-field learning networks can be implemented in a parallel, analog VLSI system. While we report early experiments on a single-chip digital system, a multiple-chip VME-based electronic system with analog I/O is being constructed for use on larger problems.

**ACKNOWLEDGMENT:**
This work has been partially supported by AFOSR contract F49620-90-C-0042, DEF.

*REFERENCES*

1. D.E. Rumelhart, G.E. Hinton, & R.J. Williams, "Learning Internal Representations by Error Propagation", in *Parallel Distributed Processing: Explorations in the Microstructure of Cognition. Vol. 1: Foundations*, D.E. Rumelhart & J.L. McClelland (eds.), MIT Press, Cambridge, MA (1986), p. 318.

2. J.J. Hopfield, "Neural Networks and Physical Systems with Emergent Collective Computational Abilities", *Proc. Natl. Acad. Sci. USA*, **79** , 2554-2558 (1982).

3. D.H. Ackley, G.E. Hinton, & T.J. Sejnowski, "A Learning Algorithm for Boltzmann Machines", *Cognitive Science* **9** (1985) pp. 147-169.

4. C. Peterson & J.R. Anderson, "A Mean Field Learning Algorithm for Neural Networks", *Complex Systems*, 1:5, 995-1019, (1987).

5. J. Alspector, B. Gupta, & R.B. Allen, "Performance of a Stochastic Learning Microchip" in *Advances in Neural Information Processing Systems 1*, D. Touretzky (ed.), Morgan-Kaufmann, Palo Alto, (1989), pp. 748-760.

6. J. Alspector, R.B. Allen, A. Jayakumar, T. Zeppenfeld, & R. Meir "Relaxation Networks for Large Supervised Learning Problems" in *Advances in Neural Information Processing Systems 3*, R.P Lippmann, J.E. Moody, & D.S. Touretzky (eds.), Morgan-Kaufmann, Palo Alto, (1991), pp. 1015-1021.

7. J. Alspector, J.W. Gannett, S. Haber, M.B. Parker, & R. Chu, "A VLSI-Efficient Technique for Generating Multiple Uncorrelated Noise Sources and Its Application to Stochastic Neural Networks", *IEEE Trans. Circuits & Systems*, **38**, 109, (Jan., 1991).

8. R.B. Allen & J. Alspector, "Learning of Stable States in Stochastic Asymmetric Networks", *IEEE Trans. Neural Networks*, **1**, 233-238, (1990).

9. J. Alspector, T. Zeppenfeld & S. Luna, "A Volatility Measure for Annealing in Feedback Neural Networks", to appear in *Neural Computation*, (1992).

## Footnotes

† Permanent address: University of California, Berkeley; EECS Dep't, Cory Hall; Berkeley, CA 94720
